# Automatic Capacity Tuning
# of Very Large VC-dimension Classifiers

**I. Guyon**
AT&T Bell Labs,
50 Fremont st., $6^{th}$ floor,
San Francisco, CA 94105
isabelle@neural.att.com

**B. Boser***
EECS Department,
University of California,
Berkeley, CA 94720
boser@eecs.berkeley.edu

**V. Vapnik**
AT&T Bell Labs,
Room 4G-314,
Holmdel, NJ 07733
vlad@neural.att.com

## Abstract

Large VC-dimension classifiers can learn difficult tasks, but are usually impractical because they generalize well only if they are trained with huge quantities of data. In this paper we show that even high-order polynomial classifiers in high dimensional spaces can be trained with a small amount of training data and yet generalize better than classifiers with a smaller VC-dimension. This is achieved with a maximum margin algorithm (the Generalized Portrait). The technique is applicable to a wide variety of classifiers, including Perceptrons, polynomial classifiers (sigma-pi unit networks) and Radial Basis Functions. The effective number of parameters is adjusted automatically by the training algorithm to match the complexity of the problem. It is shown to equal the number of those training patterns which are closest patterns to the decision boundary (supporting patterns). Bounds on the generalization error and the speed of convergence of the algorithm are given. Experimental results on handwritten digit recognition demonstrate good generalization compared to other algorithms.

## 1   INTRODUCTION

Both experimental evidence and theoretical studies [1] link the generalization of a classifier to the error on the training examples and the capacity of the classifier.

Classifiers with a large number of adjustable parameters, and therefore large capacity, likely learn the training set without error, but exhibit poor generalization. Conversely, a classifier with insufficient capacity might not be able to learn the task at all. The goal of capacity tuning methods is to find the optimal capacity which minimizes the expected generalization error for a given amount of training data.

Capacity tuning techniques include: starting with a low capacity system and allocating more parameters as needed or starting with an large capacity system and eliminating unnecessary adjustable parameters with regularization. The first method requires searching in the space of classifier structures which possibly contains many local minima. The second method is computationally inefficient since it does not avoid adjusting a large number of parameters although the effective number of parameters may be small.

With the method proposed in this paper, the capacity of some very large VC-dimension classifiers is adjusted automatically in the process of training. The problem is formulated as a quadratic programming problem which has a single global minimum. Only the effective parameters get adjusted during training which ensures computational efficiency.

## 1.1   MAXIMUM MARGIN AND SUPPORTING PATTERNS

Here is a familiar problem: Given is a limited number of training examples from two classes A and B; find the linear decision boundary which yields best generalization performance. When the training data is scarce, there exists usually many errorless separations (figure 1.1). This is especially true when the dimension of input space (i.e. the number of tunable parameters) is large compared to the number of training examples. The question arises which of these solutions to choose? The one solution that achieves the largest possible margin between the decision boundary and the training patterns (figure 1.2) is optimal in the "minimax" sense [2] (see section 2.2). This choice is intuitively justifiable: a new example from class A is likely to fall within or near the convex envelope of the examples of class A (and similarly for class B). By providing the largest possible "safety" margin, we minimize the chances that examples from class A and B cross the border to the wrong side.

An important property of the maximum margin solution is that it is only dependent upon a restricted number of training examples, called supporting patterns (or informative patterns). These are those examples which lie on the margin and therefore are closest to the decision boundary (figure 1.2). The number $m$ of linearly independent supporting patterns satisfies the inequality:

$$m \leq \min(N + 1, p). \tag{1}$$

In this inequality, $(N + 1)$ is the number of adjustable parameters and equals the Vapnik-Chervonenkis dimension (VC-dimension) [2], and $p$ is the number of training examples. In reference [3], we show that the generalization error is bounded by $m/p$ and therefore $m$ is a measure of complexity of the learning problem. Because $m$ is bounded by $p$ and is generally a lot smaller than $p$, the maximum margin solution obtains good generalization even when the problem is grossly underdetermined, i.e. the number of training patterns $p$ is much smaller than the number of adjustable parameters, $N + 1$. In section 2.3 we show that the existence of supporting patterns is advantageous for computational reasons as well.

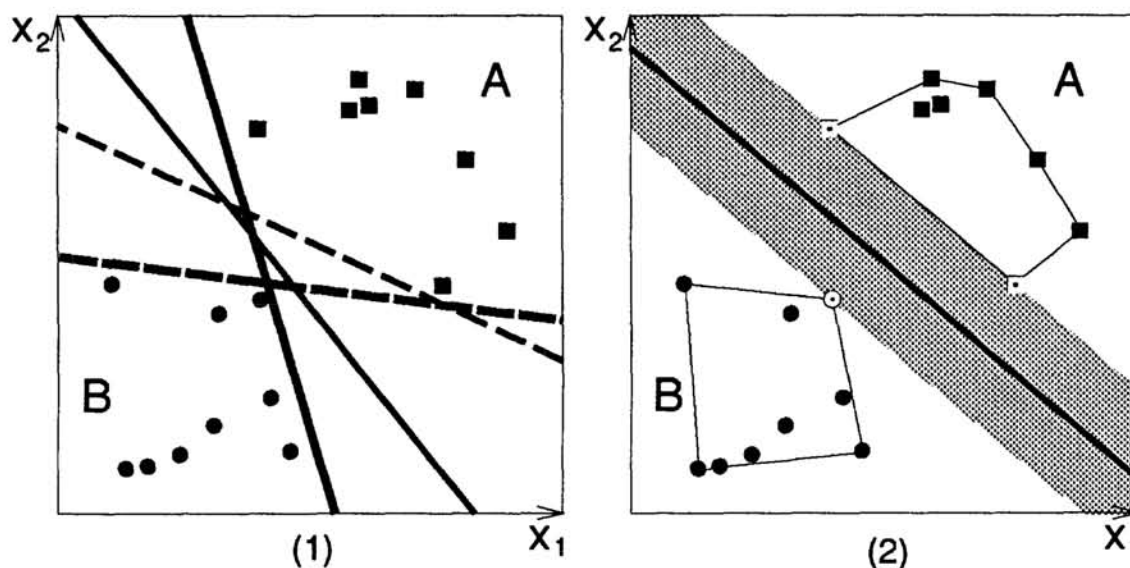

Figure 1: **Linear separations.**
(1) When many linear decision rules separate the training set, which one to choose?
(2) The maximum margin solution. The distance to the decision boundary of the
closest training patterns is maximized. The grey shading indicates the margin area
in which no pattern falls. The supporting patterns (in white) lie on the margin.

## 1.2   NON-LINEAR CLASSIFIERS

Although algorithms that maximize the margin between classes have been known
for many years [4, 2], they have for computational reasons so far been limited to the
special case of finding linear separations and consequently to relatively simple clas-
sification problems. In this paper, we present an extension to one of these maximum
margin training algorithms called the "Generalized Portrait Method" ($GP$) [2] to
various non-linear classifiers, including including Perceptrons, polynomial classifiers
(sigma-pi unit networks) and kernel classifiers (Radial Basis Functions) (figure 2).
The new algorithm trains efficiently very high VC-dimension classifiers with a huge
number of tunable parameters. Despite the large number of free parameters, the
solution exhibits good generalization due to the inherent regularization of the max-
imum margin cost function.

As an example, let us consider the case of a second order polynomial classifier. Its
decision surface is described by the following equation:

$$\sum_i w_i x_i + \sum_{i,j} w_{ij} x_i x_j + b = 0. \tag{2}$$

he $w_i$, $w_{ij}$ and $b$ are adjustable parameters, and $x_i$ are the coordinates of a pattern
**x**. If $n$ is the dimension of input pattern **x**, the number of adjustable parameters
of the second order polynomial classifier is $[n(n+1)/2]+1$. In general, the number
of adjustable parameters of a $q^{th}$ order polynomial is of the order of $N \approx n^q$.

The $GP$ algorithm has been tested on the problem of handwritten digit recognition.
The input patterns consist of $16 \times 16$ pixel images ($n = 256$). The results achieved

| q | N | DB1 (p=600) error | DB1 (p=600) <m> | DB2 (p=7300) error | DB2 (p=7300) <m> |
|---|---|---|---|---|---|
| 1 (linear) | 256 | 3.2 % | 36 | 10.5 % | 97 |
| 2 | $3 \cdot 10^4$ | 1.5 % | 44 | 5.8 % | 89 |
| 3 | $8 \cdot 10^7$ | 1.7 % | 50 | 5.2 % | 79 |
| 4 | $4 \cdot 10^9$ | | | 4.9 % | 72 |
| 5 | $1 \cdot 10^{12}$ | | | 5.2 % | 69 |

Table 1: **Handwritten digit recognition experiments.** The first database (DB1) consists of 1200 clean images recorded from ten subjects. Half of this data is used for training, and the other half is used to evaluate the generalization performance. The other database (DB2) consists of 7300 images for training and 2000 for testing and has been recorded from actual mail pieces. We use ten polynomial classification functions of order q, separating one class against all others. We list the number N of adjustable parameters, the error rates on the test set and the average number <m>of supporting patterns per separating hypersurface. The results compare favorably to neural network classifiers which minimize the mean squared error with backpropagation. For the one layer network (linear classifier),the error on the test set is 12.7 % on DB1 and larger than 25 % on DB2. The lowest error rate for DB2, 4.9 %, obtained with a forth order polynomial, is comparable to the 5.1 % error obtained with a multi-layer neural network with sophisticated architecture being trained and tested on the same data [6].

with polynomial classifiers of order q are summarized in table 1. Also listed is the number of adjustable parameters, N. This quantity increases rapidly with q and quickly reaches a level that is computationally intractable for algorithms that explicitly compute each parameter [5]. Moreover, as N increases, the learning problem becomes grossly underdetermined: the number of training patterns ($p = 600$ for DB1 and $p = 7300$ for DB2) becomes very small compared to N. Nevertheless, good generalization is achieved as shown by the experimental results listed in the table. This is a consequence of the inherent regularization of the algorithm.

An important concern is the sensitivity of the maximum margin solution to the presence of outliers in the training data. It is indeed important to remove undesired outliers (such as meaningless or mislabeled patterns) to get best generalization performance. Conversely, "good" outliers (such as examples of rare styles) must be kept. Cleaning techniques have been developed based on the re-examination by a human supervisor of those supporting patterns which result in the largest increase of the margin when removed, and thus, are the most likely candidates for outliers [3]. In our experiments on DB2 with linear classifiers, the error rate on the test set dropped from 15.2% to 10.5% after cleaning the training data (not the test data).

## 2   ALGORITHM DESIGN

The properties of the GP algorithm arise from merging two separate ideas: Training in *dual space*, and minimizing the maximum loss. For large VC-dimension classifiers ($N \gg p$), the first idea reduces the number of *effective* parameters to be actually

computed from $N$ to $p$. The second idea reduces it from $p$ to $m$.

## 2.1  DUALITY

We seek a decision function for pattern vectors $\mathbf{x}$ of dimension $n$ belonging to either of two classes A and B. The input to the training algorithm is a set of $p$ examples $\mathbf{x}_i$ with labels $y_i$:

$$(\mathbf{x}_1, y_1), \ (\mathbf{x}_2, y_2), \ (\mathbf{x}_3, y_3), \ \ldots, \ (\mathbf{x}_p, y_p) \tag{3}$$

$$\text{where} \ \begin{cases} y_k = 1 & \text{if } \mathbf{x}_k \in \text{class A} \\ y_k = -1 & \text{if } \mathbf{x}_k \in \text{class B.} \end{cases}$$

From these training examples the algorithm finds the parameters of the decision function $D(\mathbf{x})$ during a learning phase. After training, the classification of unknown patterns is predicted according to the following rule:

$$\begin{array}{ll} \mathbf{x} \in \text{A} & \text{if } D(\mathbf{x}) > 0 \\ \mathbf{x} \in \text{B} & \text{otherwise.} \end{array} \tag{4}$$

We limit ourselves to classifiers linear in their parameters, but not restricted to linear dependences in their input components, such as Perceptrons and kernel-based classifiers. Perceptrons [5] have a decision function defined as:

$$D(\mathbf{x}) = \mathbf{w} \cdot \varphi(\mathbf{x}) + b = \sum_{i=1}^{N} w_i \varphi_i(\mathbf{x}) + b, \tag{5}$$

where the $\varphi_i$ are predefined functions of $\mathbf{x}$, and the $w_i$ and $b$ are the adjustable parameters of the decision function. This definition encompasses that of polynomial classifiers. In that particular case, the $\varphi_i$ are products of components of vector $\mathbf{x}$(see equation 2). Kernel-based classifiers, have a decision function defined as:

$$D(\mathbf{x}) = \sum_{k=1}^{p} \alpha_k K(\mathbf{x}_k, \mathbf{x}) + b, \tag{6}$$

The coefficients $\alpha_k$ and the bias $b$ are the parameters to be adjusted and the $\mathbf{x}_k$ are the training patterns. The function $K$ is a predefined kernel, for example a potential function [7] or any Radial Basis Function (see for instance [8]).

Perceptrons and RBF's are often considered two very distinct approaches to classification. However, for a number of training algorithms, the resulting decision function can be cast either in the form of equation (5) or (6). This has been pointed out in the literature for the Perceptron and potential function algorithms [7], for the polynomial classifiers trained with pseudo-inverse [9] and more recently for regularization algorithms and RBF's [8]. In those cases, Perceptrons and RBF's constitute *dual* representations of the same decision function.

The duality principle can be understood simply in the case of Hebb's learning rule. The weight vector of a linear Perceptron ($\varphi_i(\mathbf{x}) = x_i$), trained with Hebb's rule, is simply the average of all training patterns $\mathbf{x}_k$, multiplied by their class membership polarity $y_k$:

$$\mathbf{w} = \frac{1}{p} \sum_{k=1}^{p} y_k \mathbf{x}_k \ .$$

Substituting this solution into equation (5), we obtain the dual representation

$$D(\mathbf{x}) = \mathbf{w} \cdot \mathbf{x} + b = \frac{1}{p} \sum_{k=1}^{p} y_k \, \mathbf{x}_k \cdot \mathbf{x} + b \;.$$

The corresponding kernel classifier has kernel $K(\mathbf{x}, \mathbf{x}') = \mathbf{x} \cdot \mathbf{x}'$ and the dual parameters $\alpha_k$ are equal to $(1/p)y_k$.

In general, a training algorithm for Perceptron classifiers admits a dual kernel representation if its solution is a linear combination of the training patterns in $\varphi$-space:

$$\mathbf{w} = \sum_{k=1}^{p} \alpha_k \varphi(\mathbf{x}_k) \;. \tag{7}$$

Reciprocally, a kernel classifier admits a dual Perceptron representation if the kernel function possesses a finite (or infinite) expansion of the form:

$$K(\mathbf{x}, \mathbf{x}') = \sum_{i} \varphi_i(\mathbf{x}) \, \varphi_i(\mathbf{x}') \;. \tag{8}$$

Such is the case for instance for some symmetric kernels [10]. Examples of kernels that we have been using include

$$
\begin{aligned}
K(\mathbf{x}, \mathbf{x}') &= (\mathbf{x} \cdot \mathbf{x}' + 1)^q && \text{(polynomial of order q),} \\
K(\mathbf{x}, \mathbf{x}') &= \tanh(\gamma \, \mathbf{x} \cdot \mathbf{x}') && \text{(neural units),} \\
K(\mathbf{x}, \mathbf{x}') &= \exp(\gamma \, \mathbf{x} \cdot \mathbf{x}') - 1 && \text{(exponential),} \\
K(\mathbf{x}, \mathbf{x}') &= \exp(-\|\mathbf{x} - \mathbf{x}'\|^2/\gamma) && \text{(gaussian RBF),} \\
K(\mathbf{x}, \mathbf{x}') &= \exp(-\|\mathbf{x} - \mathbf{x}'\|/\gamma) && \text{(exponential RBF),} \\
K(\mathbf{x}, \mathbf{x}') &= (\mathbf{x} \cdot \mathbf{x}' + 1)^q \exp(-\|\mathbf{x} - \mathbf{x}'\|/\gamma) && \text{(mixed polynomial \& RBF).}
\end{aligned}
\tag{9}
$$

These kernels have positive parameters (the integer $q$ or the real number $\gamma$) which can be determined with a Structural Risk Minimization or Cross-Validation procedure (see for instance [2]). More elaborate kernels incorporating known invariances of the data could be used also.

The *GP* algorithm computes the maximum margin solution in the kernel representation. This is crucial for making the computation tractable when training very large VC-dimension classifiers. Training a classifier in the kernel representation is computationally advantageous when the dimension $N$ of vectors $\mathbf{w}$ (or the VC-dimension $N + 1$) is large compared to the number of parameters $\alpha_k$, which equals the number of training patterns $p$. This is always true if the kernel function possesses an infinite expansions (8). The experimental results listed in table 1 indicate that this argument holds in practice even for low order polynomial expansions when the dimension $n$ of input space is sufficiently large.

## 2.2   MINIMIZING THE MAXIMUM LOSS

The margin, defined as the Euclidean distance between the decision boundary and the closest training patterns in $\varphi$-space can be computed as

$$M = \min_{k} \frac{y_k D(\mathbf{x}_k)}{\|\mathbf{w}\|} \;. \tag{10}$$

The goal of the maximum margin training algorithm is to find the decision function $D(\mathbf{x})$ which maximizes $M$, that is the solution of the optimization problem

$$\max_{\mathbf{w}} \min_{k} \frac{y_k D(\mathbf{x}_k)}{\|\mathbf{w}\|} . \tag{11}$$

The solution $\mathbf{w}$ of this problem depends only on those patterns which are on the margin, i.e. the ones that are closest to the decision boundary, called supporting patterns. It can be shown that $\mathbf{w}$ can indeed be represented as a linear combination of the supporting patterns in $\varphi$-space [4, 2, 3] (see section 2.3).

In the classical framework of loss minimization, problem 11 is equivalent to minimizing (over $\mathbf{w}$) the maximum loss. The loss function is defined as

$$l(\mathbf{x}_k) = -y_k D(\mathbf{x}_k)/\|\mathbf{w}\|.$$

This "minimax" approach contrasts with training algorithms which minimize the average loss. For example, backpropagation minimizes the mean squared error (MSE), which is the average of

$$l(\mathbf{x}_k) = (D(\mathbf{x}_k) - y_k)^2 .$$

The benefit of minimax algorithms is that the solution is a function only of a restricted number of training patterns, namely the supporting patterns. This results in high computational efficiency in those cases when the number $m$ of supporting patterns is small compared to both the total number of training patterns $p$ and the dimension $N$ of $\varphi$-space.

## 2.3    THE GENERALIZED PORTRAIT

The $GP$ algorithm consists in formulating the problem 11 in the dual $\alpha$-space as the quadratic programming problem of maximizing the cost function

$$J(\alpha, b) = \sum_{k=1}^{p} \alpha_k (1 - by_k) - \frac{1}{2} \alpha \cdot H \cdot \alpha,$$

under the constrains $\alpha_k > 0$ [4, 2]. The $p \times p$ square matrix $H$ has elements:

$$H_{kl} = y_k y_l K(\mathbf{x}_k, \mathbf{x}_l).$$

where $K(\mathbf{x}, \mathbf{x}')$ is a kernel, such as the ones proposed in (9), which can be expanded as in (8). Examples are shown in figure 2. $K(\mathbf{x}, \mathbf{x}')$ is not restricted to the dot product $K(\mathbf{x}, \mathbf{x}') = \mathbf{x} \cdot \mathbf{x}'$ as in the original formulation of the $GP$ algorithm [2].

In order for a unique solution to exist, $H$ must be positive definite. The bias $b$ can be either fixed or optimized together with the parameters $\alpha_k$. This case introduces another set of constraints: $\sum_k y_k \alpha_k = 0$ [4].

The quadratic programming problem thus defined can be solved efficiently by standard numerical methods [11]. Numerical computation can be further reduced by processing iteratively small chunks of data [2]. The computational time is linear the dimension $n$ of $\mathbf{x}$-space (not the dimension $N$ of $\varphi$-space) and in the number $p$ of training examples and polynomial in the number $m < \min(N + 1, p)$ of supporting

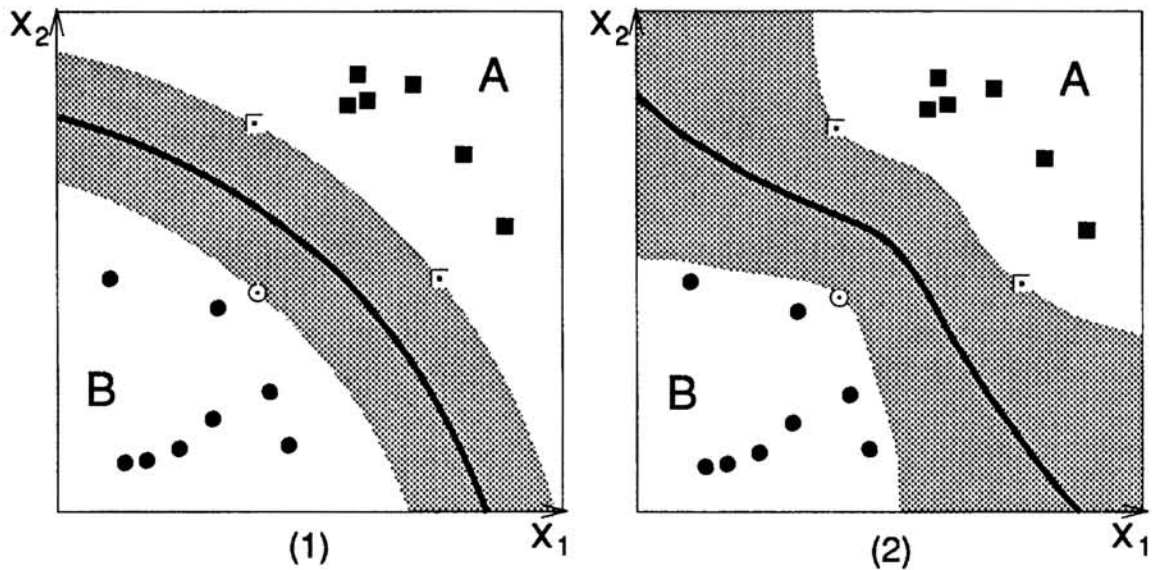

Figure 2: **Non-linear separations.**
Decision boundaries obtained by maximizing the margin in $\varphi$-space (see text). The grey shading indicates the margin area projected back to **x**-space. The supporting patterns (white) lie on the margin. (1) Polynomial classifier of order two (sigma-pi unit network), with kernel $K(\mathbf{x}, \mathbf{x}') = (\mathbf{x} \cdot \mathbf{x}' + 1)^2$. (2) Kernel classifier (RBF) with kernel $K(\mathbf{x}, \mathbf{x}) = (\exp -\|\mathbf{x} - \mathbf{x}'\|/10)$.

patterns. It can be theoretically proven that it is a polynomial in $m$ of order lower than 10, but experimentally an order 2 was observed.

Only the supporting patterns appear in the solution with non-zero weight $\alpha_k$:

$$D(\mathbf{x}) = \sum_k y_k \alpha_k K(\mathbf{x}_k, \mathbf{x}) + b, \qquad \alpha_k \geq 0 \ . \tag{12}$$

Substituting (8) in $D(\mathbf{x})$, we obtain:

$$\mathbf{w} = \sum_k y_k \alpha_k \varphi(x_k) \ . \tag{13}$$

Using the kernel representation, with a factorized kernel (such as 9), the classification time is linear in $n$ (not $N$) and in $m$ (not $p$).

## 3   CONCLUSIONS

We presented an algorithm to train in high dimensional spaces polynomial classifiers and Radial Basis functions which has remarquable computational and generalization performances. The algorithms seeks the solution with the largest possible margin on both side of the decision boundary. The properties of the algorithm arise from the fact that the solution is a function only of a small number of supporting patterns, namely those training examples that are closest to the decision boundary. The generalization error of the maximum margin classifier is bounded by the ratio

of the number of linearly independent supporting patterns and the number of training examples. This bound is tighter than a bound based on the VC-dimension of the classifier family. For further improvement of the generalization error, outliers corresponding to supporting patterns with large $\alpha_k$ can be eliminated automatically or with the assistance of a supervisor. This feature suggests other interesting applications of the maximum margin algorithm for database cleaning.

## Acknowledgements

We wish to thank our colleagues at UC Berkeley and AT&T Bell Laboratories for many suggestions and stimulating discussions. Comments by L. Bottou, C. Cortes, S. Sanders, S. Solla, A. Zakhor, are gratefully acknowledged. We are especially indebted to R. Baldick and D. Hochbaum for investigating the polynomial convergence property, S. Hein for providing the code for constrained nonlinear optimization, and D. Haussler and M. Warmuth for help and advice regarding performance bounds.

## Footnotes

*Part of this work was done while B. Boser was at AT&T Bell Laboratories. He is now at the University of California, Berkeley.

## References

[1] I. Guyon, V. Vapnik, B. Boser, L. Bottou, and S.A. Solla. Structural risk minimization for character recognition. In J. Moody and et al., editors, *NIPS 4*, San Mateo CA, 1992. IEEE, Morgan Kaufmann.

[2] V.N. Vapnik. *Estimation of dependences based on empirical data*. Springer, New York, 1982.

[3] B. Boser, I. Guyon, and V. Vapnik. An training algorithm for optimal margin classifiers. In *Fifth Annual Workshop on Computational Learning Theory*, pages 144–152, Pittsburgh, July 1992. ACM.

[4] P. F. Lambert. Designing patterns recognizers with extremal paradigm information. In Watanabe S., editor, *Methodologies of Pattern Recognition*, pages 359–391. Academic Press, 1969.

[5] R.O. Duda and P.E. Hart. *Pattern Classification And Scene Analysis*. Wiley and Son, 1973.

[6] Y. Le Cun, B. Boser, J. S. Denker, D. Henderson, R. E. Howard, W. Hubbard, and L. D. Jackel. Back-propagation applied to handwritten zipcode recognition. *Neural Computation*, 1(4):541–551, 1989.

[7] M.A. Aizerman, E.M. Braverman, and L.I. Rozonoer. Theoretical foundations of the potential function method in pattern recognition learning. *Automation and Remote Control*, 25:821–837, 1964.

[8] T. Poggio and F. Girosi. Regularization algorithms for learning that are equivalent to multilayer networks. *Science*, 247:978 – 982, February 1990.

[9] T. Poggio. On optimal nonlinear associative recall. *Biol. Cybern.*, 19:201, 1975.

[10] G.F Roach. *Green's Functions*. Cambridge University Press, Cambridge, 1982 (second ed.).

[11] D. Luenberger. *Linear and Non-linear Programming*. Addidon Wesley, 1984.